# Stress, noradrenaline, and realistic prediction of mouse behaviour using reinforcement learning

**Gediminas Lukšys[1,2], Carmen Sandi[2], Wulfram Gerstner[1]**
[1]Laboratory of Computational Neuroscience
[2]Laboratory of Behavioural Genetics
Ecole Polytechnique Fédérale de Lausanne (EPFL)
Lausanne, CH-1015, Switzerland
{gediminas.luksys,carmen.sandi,wulfram.gerstner}@epfl.ch

## Abstract

Suppose we train an animal in a conditioning experiment. Can one predict how a given animal, under given experimental conditions, would perform the task? Since various factors such as stress, motivation, genetic background, and previous errors in task performance can influence animal behaviour, this appears to be a very challenging aim. Reinforcement learning (RL) models have been successful in modeling animal (and human) behaviour, but their success has been limited because of uncertainty as to how to set meta-parameters (such as learning rate, exploitation-exploration balance and future reward discount factor) that strongly influence model performance. We show that a simple RL model whose meta-parameters are controlled by an artificial neural network, fed with inputs such as stress, affective phenotype, previous task performance, and even neuromodulatory manipulations, can successfully predict mouse behaviour in the "hole-box" - a simple conditioning task. Our results also provide important insights on how stress and anxiety affect animal learning, performance accuracy, and discounting of future rewards, and on how noradrenergic systems can interact with these processes.

## 1   Introduction

Animal behaviour is guided by rewards that can be received in different situations and by modulatory factors, such as stress and motivation. It is known that acute stress can affect learning and memory by modulating plasticity through stress hormones and neuromodulators [1, 2, 3], but their role in high-level processes such as learning, memory, and action selection is not well understood. A number of interesting conceptual and computational models have been proposed relating neuromodulatory systems, cognitive processes, and abstract statistical quantities characterizing the environment [4, 5]. While such models provide great mechanistic insights, they alone are often unable to accurately predict animal behaviour in a realistic situation due to a great number of diverse modulatory factors. Stress [2], genotype [6], affective traits such as anxiety and impulsivity [7], motivation [8], and evaluation of performance errors [9] can all influence individual performance in any single task, yet it may prove difficult and inefficient to explicitly model each factor in order to accurately predict animal behaviour. Instead, we propose a method which could account for the influence of arbitrary modulatory factors on behaviour as control parameters of a general behavioural model.

In modeling reward-based behavioural learning, approaches based on the formal theory of reinforcement learning (RL) have been the most successful. The basic idea of RL is that animals (or artificial agents) select their actions based on predicted future rewards that could be acquired upon taking these actions. The expected values of future rewards for different actions (Q-values) can be gradually learned by observing rewards received under different state-action combinations. An efficient

way to do this is temporal difference (TD) learning [10], which uses an error signal that correlates with the activity of dopaminergic neurons in the Substantia Nigra [11]. TD models have been successfully applied to explain a wide range of experimental data, including animal conditioning [8], human decision-making [12], and even addiction [13].

Learning and action selection in TD models can be strongly influenced by the choice of model meta-parameters such as the learning rate, the future reward discounting, and the exploitation-exploration balance. While in most modeling studies they have received relatively little attention, it has been proposed that RL meta-parameters are related to specific neuromodulators - noradrenaline, serotonin, acetylcholine [14], and to neural activity occurring in different brain regions - notably amygdala, striatum, and anterior cingulate [15]. Modulatory factors such as stress, anxiety, and impulsivity often act through the same brain systems, which suggests that in RL models their effects could be expressed through changes in meta-parameter values.

In the present study, we tested mouse behaviour in a simple conditioning task - the hole-box, and showed how various modulatory factors could control a simple RL model to accurately predict animal behaviour. We used food deprived mice of two genetic strains - 'calm' C57BL/6 and 'anxious' DBA/2 [6], half of which were exposed to an additional stressor - sitting on an elevated platform - before each experimental session. We formalized animal behaviour using a simple RL model, and trained an artificial neural network that could control RL meta-parameters using information about stress, motivation, individual affective traits, and previous learning success. We demonstrate that such model can successfully predict mouse behaviour in the hole-box task and that the resulting model meta-parameters provide useful insights into how animals adjust their performance throughout the course of a learning experience, and how they respond to stressors and motivational demands. Finally, using systemic manipulations of the noradrenergic system we show how noradrenaline interacts with stress and anxiety in regulating performance accuracy and temporal discounting.

## 2  Description of the hole-box experiment

In our hole-box experiments, we used 64 male mice (32 of C57BL/6 strain and 32 of DBA/2 strain) that were 10-week old at the beginning of the experiment. During an experimental session, each animal was placed into the hole-box (Figure 1a). The mice had to learn to make a nose poke into the hole upon the onset of lights and not to make it under the condition of no light. After a response to light, the animals (which were food deprived to 87.3+/-1.0% of their initial weight) received a reward in form of a food pellet (Figure 1b). The inter-trial interval (ITI) between subsequent trials was varying: the probability of a new trial during each 0.5 sec long time step was 1/30, resulting in the average ITI of 15 sec. The total session duration was 500 sec, equivalent to 1000 time steps.

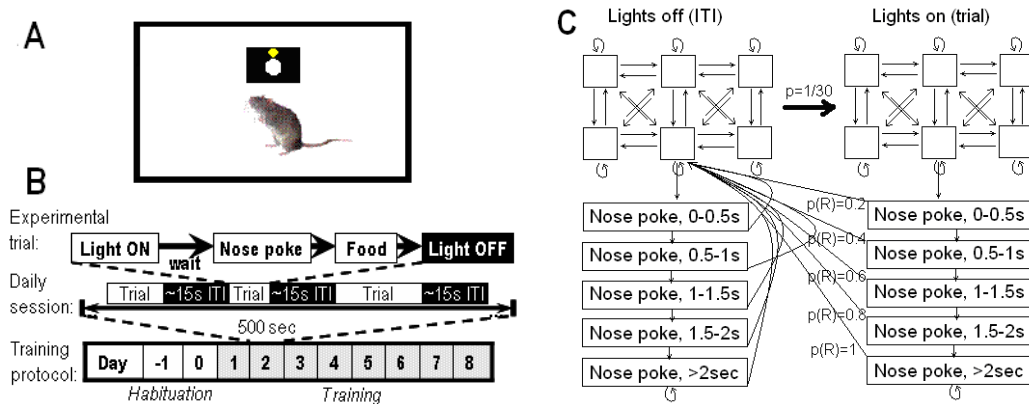

Figure 1: **a.** Scheme of the hole-box. **b.** Protocol of the hole-box experiment. **c.** Hole-box state-action chart. Rectangles are states, thin arrows are actions.

During 2 days of habituation (when the food delivery was not paired with light) the mice learned that food could be delivered from the boxes. After this, they were trained for 8 consecutive days, during

which half of the mice were exposed to extrinsic stress (30min on the elevated platform) before each training session. On training days 3, 6, and 8, animals have been injected i.p. (5 ml/kg, 30 min before the experimental session) with either saline (1/2 of mice), or adrenergic alpha-2 agonist clonidine (1/4 of mice, 0.05 mg/kg) that reduces brain noradrenaline levels, or adrenergic alpha-2 antagonist yohimbine (1/4 of mice, 1 mg/kg) that increases brain noradrenaline levels. Mice of each strain were treated equivalently with respect to pharmacological and stress conditions. Stress and pharmacological treatment groups were the same during all training days.

## 3 Challenges of behavioural analysis

To quantify animal performance in the hole-box experiment, we used 7 different performance measures (PMs). These were behavioural statistics, calculated for each daily session: number of trials (within 500 sec), number of ITI pokes, mean response time (after light onset), mean nose poke duration, number of uneaten food pellets, "TimePreference"[1], and "DurationPreference"[2]. Different PMs reflected different aspects of behaviour - learning to respond, associating responses with light, overcoming anxiety to make sufficiently long nose pokes, etc. For this reason, during the process of learning PMs exhibited a variety of dynamics: slowly increasing numbers of trials, rapidly decreasing mean response times, first increasing and later decreasing numbers of ITI pokes.

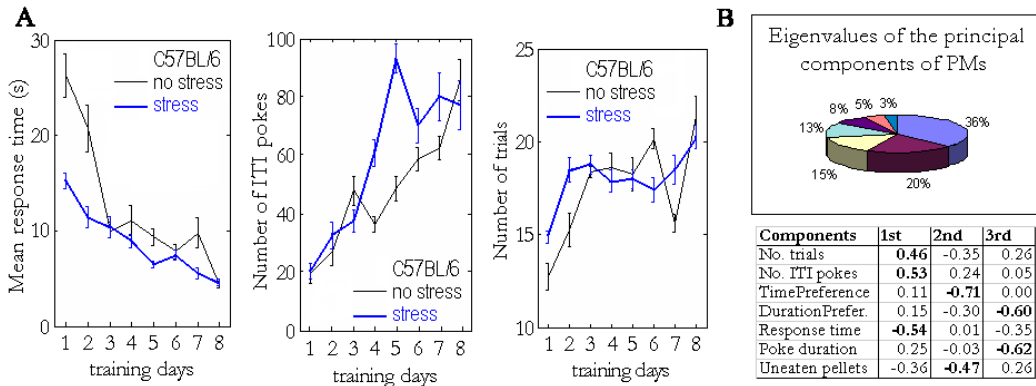

Figure 2: **a.** Development of selected PMs with learning for C57BL/6 mice. **b.** Results of the PCA applied for all PMs: eigenvalues and loadings for the first 3 components.

When comparing the PMs between different experimental groups (Figure 2a), it is often hard to interpret the differences, as each PM describes an unknown mixture of cognitive processes such as learning, memory, performance intensity and accuracy. In some cases, performing a principal component analysis (PCA) or similar tools may be suitable for reducing the behavioural measures to few main components that could be easily interpreted [16]. However, more often that is not the case - for instance, in our experiment, the 3 principal components are not sufficient to explain even 75% of the variation, and the composition of the components is not easy to interpret (Figure 2b). As an alternative to conventional behavioural analysis, we propose that a computational model of behaviour, based on reinforcement learning, could be sufficiently flexible to fit a wide range of behavioural effects, and in contrast to the PMs, RL meta-parameters could be easily interpreted in cognitive terms.

## 4 Modeling the hole-box using reinforcement learning

We used a simple temporal difference RL model to formalize the behaviour. Conceptually, the model had 4 states: [*ITI*, *trial*] x [*animal outside*, *making a nose poke*], and 2 actions: *move* (in or out) and *stay*. However, to make model's performance realistic several extensions had to be introduced (Figure 1c). First of all, the state *animal outside* was divided into 6 states corresponding

to different places in the box which the animal could occupy, adding additional actions for the transitions between these new states (moving around the box). Secondly, we observed that when our animals made too short trial responses (with nose poke duration under 0.5 sec), they often could not pick up the delivered food. Conversely, when the nose pokes were longer than 1.5 sec, animals nearly always managed to pick up the delivered food immediately. To account for this, the state *making a nose poke* was divided into 5 states, representing different nose poke durations, with the increasing probability of picking up the reward (to keep things simple, we chose a linear increase: from $p = 0.2$ for the first state to $p = 1.0$ for the fifth). Note that a food pellet is delivered at the start of each trial response, irrespectively of whether the animal picks it up during that nose poke or not. Unconsumed pellets could be eaten during later (sufficiently long) ITI nose pokes.

The Q-values, defined as $Q(s_t, a_t) = E[r(t) + \gamma r(t+1) + \gamma^2 r(t+2) + ...|s_t, a_t]$, were updated based on the temporal difference error:

$$\Delta Q(s_t, a_t) = \alpha[r(t) + \gamma Q(s_{t+1}, a_{t+1}) - Q(s_t, a_t)], \tag{1}$$

where $r(t)$ is the reward at time $t$, $s_t$ the state, $a_t$ the action, $\alpha$ the learning rate, and $\gamma$ the future reward discount factor. High $\gamma$ values (close to 1) signified that future rewards were given high weight, while low $\gamma$ values (0-0.5) meant that immediate rewards were preferred. Actions were chosen probabilistically, based on Q-values and the exploitation factor $\beta$, as follows:

$$p(a_i|s) = \exp(\beta Q(s, a_i))/ \sum_{k \in A(s)} \exp(\beta Q(s, a_k))) \tag{2}$$

where A(s) are actions available at state s. Low $\beta$ values implied that the actions were being chosen more or less randomly (exploration), while high $\beta$ values strongly biased the choice towards the action(s) with the highest Q-value (exploitation). Q-values were initialized as zeros before the first training day, and the starting state was always *ITI / outside, near the hole*.

## 5   Predicting mouse behaviour using dynamic control of model meta-parameters

To compare the model with animal behaviour we used the following goodness-of-fit function [17]:

$$\chi^2 = \sum_{k=1}^{N_{\mathrm{PM}}} (\mathrm{PM}_k^{\mathrm{exp}} - \mathrm{PM}_k^{\mathrm{mod}}(\alpha, \beta, \gamma))^2/(\sigma_k^{\mathrm{exp}})^2 \ , \tag{3}$$

where $\mathrm{PM}_k^{\mathrm{exp}}$ and $\mathrm{PM}_k^{\mathrm{mod}}$ are the PMs calculated for each animal and the model, respectively, and $N_{\mathrm{PM}} = 7$ is the number of the PMs. $\mathrm{PM}_k^{\mathrm{mod}}(\alpha, \beta, \gamma)$ were calculated after simulation of one session (averaged over multiple runs) with fixed values of the meta-parameters. To evaluate whether our model is sufficiently flexible to fit a wide range of animal behaviours (including effects of stress, strain, and noradrenaline), we performed an estimation procedure of daily meta-parameters. Using stochastic gradient ascent from multiple starting points, we minimized (3) with respect to $\alpha, \beta, \gamma$ for each session separately by systematically varying the meta-parameters in the following ranges: $\alpha, \gamma \in [0.03, 0.99]$ and $\beta \in [10^{-1}, 10^{1.5}]$. To evaluate how well the model fits the experimental data we used $\chi^2$-test with $\nu = N_{\mathrm{PM}} - 3$ degrees of freedom (since our model has 3 free parameters). The $P(\chi^2, \nu)$ value, defined as the probability that a realization of a chi-square-distributed random variable would exceed $\chi^2$ by chance, was calculated for each session separately. Generally, values of $P(\chi^2, \nu) > 0.01$ correspond to a fairly good model [17].

Even if our RL model with estimated meta-parameters is capable of reproducing behaviour of different experimental groups in the hole-box, this does not tell us how, *given a new animal* in *an arbitrary experimental condition*, we should set daily meta-parameters to predict its behaviour. However, information about animal's affective phenotype, its experimental condition, and recent task performance may be helpful in determining these meta-parameter settings, and thus, predicting behaviour. For this purpose, we trained an artificial neural network (NN) model (Figure 3b), whose outputs would be the predicted values of $\alpha$, $\beta$, and $\gamma$. The inputs of the model included the following information: animal's genetic strain (0 for C57BL/6, 1 for DBA/2), its anxiety (% of time it spends in the center of the *open field* - a separate experiment for characterization of affective traits), its novelty response (% of time it spends in the center of the field once a novel object is introduced there),

stress prior to a training session (0 or 1), motivation (% of initial weight, correlating with hunger), noradrenergic manipulation (-1 for NA reduction, 1 for NA increase, and 0 for control), and two important measures describing performance on the previous day - a number of food pellets eaten ('rewards'), and a number of nose pokes during which no food was consumed ('misses'). Our NN had merely 4 hidden layer "neurons" (to prevent from over-fitting, as we only had 762 samples of data for training and validation). Its target outputs were the daily estimated meta-parameter sets, and after normalizing inputs and targets to zero mean and unit variance, the network was trained (100 times) using the Levenberg-Marquardt method [18]. Because of the normalization, the resulting mean square errors (MSEs) directly indicated how much variance in the meta-parameters could not be explained by the NN.

Using 10 trained networks with lowest MSEs, we performed simulations to analyze how much different input factors affect each meta-parameter. For this purpose we simulated the NN $10^6$ times, linearly varying 1 or 2 selected inputs, while all the remaining inputs would be given random values with zero mean and unit variance. Then we could plot mean resulting meta-parameter values corresponding to different values of the selected inputs. The range of meta-parameter variation and relative noise in such plots indicated how strongly the selected inputs (compared to other inputs) influenced the resulting meta-parameters. Finally, to predict the performance of selected animals and the differences between experimental groups, we simulated the NN with input values of each animal and analyzed the resulting meta-parameters.

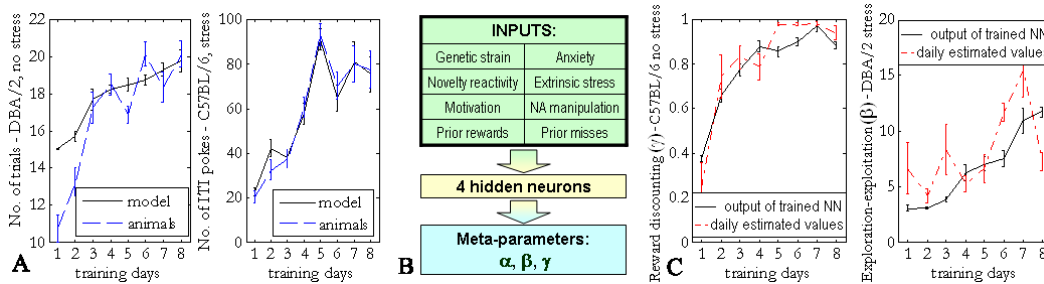

Figure 3: **a.** Comparison of model performance and animal behaviour. **b.** Scheme of the NN model. **c.** Comparison of daily estimated meta-parameters and outputs of the trained NN model. In **a** and **c** arbitrary performance measures and experimental groups were selected for comparison.

# 6 Results

The results of daily meta-parameter estimation indicated a good fit between the model and animal performance (Figure 3a). The condition $P(\chi^2, \nu) > 0.01$ was satisfied for 92% of estimated parameter sets. The mean $\chi^2$ value was $\langle\chi^2\rangle = 5.4$, or only $\langle\chi^2\rangle = 0.77$ per PM.

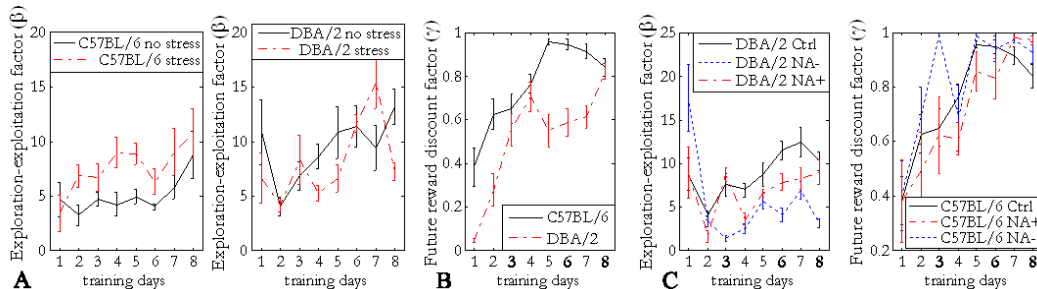

Figure 4: Estimated daily meta-parameter values and differences between experimental conditions. **a.** Exploitation factors $\beta$, strain, and stress. **b.** Reward discount factors $\gamma$ and mouse strain. **c.** Effects of noradrenergic manipulations (on days 3, 6, and 8).

Meta-parameters, estimated for each daily session, indicated interesting dynamics as well as some profound differences depending on stress condition, animal's strain, and noradrenergic manipulation. During the process of learning, estimated exploitation-exploration factors $\beta$ and future reward discount factors $\gamma$ showed progressive increase (Figure 4a,b; regression $p < 0.001$), meaning that the better animals learn the task - the more accurately they use their knowledge for selecting actions, and the longer time horizon they can take into account. In addition, extrinsic stress increases exploitation factors $\beta$ for calm C57BL/6 mice (ANOVA $p < 0.01$) but not for anxious DBA/2 mice (Figure 4a). Reward discount factors $\gamma$ were higher for C57BL/6 mice (Figure 4b, ANOVA $p < 0.001$), indicating that anxious DBA/2 mice act more impulsively. Dynamics of the learning rates and effects of stress on future reward discounting showed certain trends, however, for these daily estimated values they were not significant. For the pharmacological manipulations, two results were significant (Figure 4c): a decrease in noradrenaline led to reduced exploitation factors for the anxious DBA/2 mice (ANOVA $p < 0.001$), and to increased reward discount factors for C57BL/6 mice (on day 3, t-test $p < 0.01$), suggesting that decreasing NA levels counteracts anxiety and impulsivity.

A problem of daily estimated meta-parameters is their excessive flexibility, allowing them to follow everyday ups and downs of individual animal behaviour, many of which happen because of factors unknown to the experimenter. This "noise" often makes it difficult to see the effects that known factors (such as stress and strain) have on meta-parameter dynamics. Results of the trained NN model for prediction of daily meta-parameters indicated that only about 25% of their variation could be explained. However, the resulting meta-parameter averages for experimental groups indicated a very good fit with estimated daily meta-parameters (Figure 3c). It is also evident that different meta-parameters can be predicted to a different extent: for the learning rates only a small part of variation can be explained ($MSE(\alpha) = 0.92$), while for exploitation and reward discount factors - a substantial part ($MSE(\beta) = 0.72, MSE(\gamma) = 0.62$), showing that their values are more reliable and more sensitive to modulatory influences. The comparison of NN training and validation errors (Figure 5a) indicated that the effects of over-fitting were negligible.

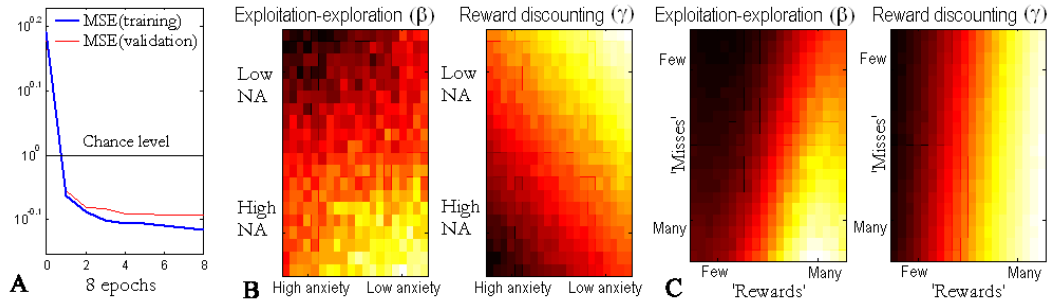

Figure 5: **a.** Typical training and validation errors for the NN model. **b.** Model simulations: interactions between anxiety and noradrenaline in affecting exploitation factors $\beta$ and reward discount factors $\gamma$. **c.** Model simulations: interactions between *rewards* and *misses* in task performance. In **b** and **c** light colors represent high meta-parameter values, dark colors - low values.

The meta-parameter prediction model allows us to analyze how (and how much) each modulatory factor affects meta-parameters and what the interactions between factors are. This is particularly useful for studying possibly non-linear interactions between continuous-valued factors, such as anxiety, motivation, and previous task performance. Results in Figure 5b,c describe such interactions. The level of noise in the color plots indicate that previous task performance (Fig. 5c) has a relatively strong influence on meta-parameters, compared to that of anxiety (Fig. 5b). Future reward discounting is mainly affected by received *rewards*, while for exploitation factors *misses* also have a significant effect, supporting an observation that well trained animals (who receive many rewards and make few misses) decrease their effort to perform quickly and accurately (Fig. 5c). Finally, anxiety and high noradrenaline levels act additively in lowering the reward discount factors, while their effects on exploitation factors are more complex: for calm animals NA increase leads to higher exploitation, but for highly anxious animals (whose NA levels are already presumably high) increasing NA does not improve their performance accuracy (Fig. 5b).

When comparing meta-parameter averages between various experimental conditions, the output of the NN model fits well the daily estimated values (Figure 3c), however, the dynamics become much smoother and the error bars - much smaller, since they account only for known factors, included in the NN input. While *all* meta-parameter effects observed when comparing daily estimated values are reproduced, excluding unpredicted variability makes some additional effects statistically significant. For instance, it is evident (Figure 6a) that extrinsic stress decreases future reward discount factors for the DBA/2 mice (ANOVA $p < 0.01$) and that the learning rates slightly decrease with learning, particularly for the C57BL/6 mice (regression $p < 0.01$). The effects of the pharmacological manipulations of the noradrenergic system have been "denoised" as well, and several additional effects become evident (Figure 6b). For C57BL/6 mice, stress plays an important role in modulating effects of NA: non-stressed mice increase their exploitation upon increased NA level (ANOVA $p < 0.01$), and slightly decrease it upon decreased NA levels. Stressed mice do not show significant changes in exploitation factors. For DBA/2 mice, stimulating noradrenergic function does not lead to higher exploitation factors (similarly to stressed C57BL/6 mice), but their future reward discounting is sensitive to NA changes - the lower NA, the higher their $\gamma$ values (ANOVA, $p < 0.01$).

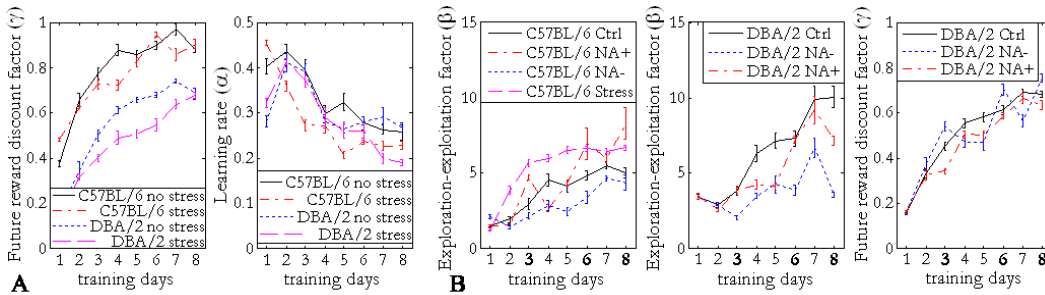

Figure 6: "Denoised" meta-parameters: outputs of the trained neural network model. Several additional differences between experimental conditions become evident. **a.** Meta-parameters, stress, and strain. **b.** Effects of noradrenergic manipulations (on days 3, 6, and 8).

## 7    Discussion

In this paper, we demonstrated that a simple RL model, whose parameters are controlled by a neural network that uses the information about various modulatory influences, can successfully predict mouse behaviour in the hole-box conditioning task. Compared to the conventional performance measures, the resulting meta-parameters of our model showed more pronounced effects between experimental groups and they have the additional advantage of being easier to relate to cognitive processes. Moreover, the results of pharmacological manipulations provided supporting evidence that RL meta-parameters are indeed related to neuromodulators such as noradrenaline.

The progressive increase of exploitation factors $\beta$ and the decrease of learning rates $\alpha$ are consistent with how the meta-parameters of artificial agents should presumably be controlled to achieve optimal performance [14]. The increase in reward discount factors $\gamma$ may have fundamental reasons too, e.g. when exposed to a new environment, hungry animals may become anxious about the uncertainty in the situation (whether they will be able to find food to survive), which makes them prefer immediate rewards to delayed ones. However, it may also be related to the specific reward structure in the model. In order to stay in the hole for longer than 1 time step (and thus have a higher chance to pick up the food) $\gamma$ values should be much larger than $0.5$. In addition, to avoid making unnecessary ITI pokes (given that food is usually picked up during the trial response) $\gamma$ values close to $1.0$ are necessary. For this reason, animal behavioural dynamics (e.g. when the mice start making sufficiently long nose pokes, and when, if at all, they learn to avoid making ITI pokes) could determine (or be determined by) the prevailing dynamics of $\gamma$-s.

Our specific results provide insights into biological mechanisms of stress, anxiety, behavioural performance, and how they relate to formal RL quantities. Stress increased performance accuracy ($\beta$ factors) for the calm C57BL/6 mice, but not for the anxious DBA/2 mice. Similarly, increasing noradrenaline levels had a positive effect on $\beta$-s only for the non-stressed C57BL/6 mice, but not for

the other groups, while decreasing NA levels had the strongest negative effect on $\beta$-s for the anxious DBA/2 mice. This suggests that within a certain range (which is dependent on animal's anxiety) performance accuracy is determined by NA level. Outside this range, NA effects get saturated or may even get reversed, as suggested by the inverse-U-shaped-relation theory of arousal/stress effects on cognition [4]. The effects of stress, strain, and NA on future reward discounting indicate that stress, high anxiety, and elevated noradrenaline are all detrimental for learning delayed future rewards. However, since the effects of NA and stress on reward discount factors are more pronounced for DBA/2 mice, $\gamma$-s might be sensitive to noradrenaline at higher levels than $\beta$-s are. It is also likely that serotonin, mPFC, and other brain systems often implicated in processing of delayed rewards [15, 19] may be interacting with stress and NA in controlling future reward discounting.

Although the basis of our hole-box behavioural prediction is a simple RL model with discrete states and actions, it is not obvious that such a model could predict animal behaviour in other significantly more complex tasks. However, even in more complex models (involving continuous state-action spaces, episodic memories, etc.), a RL-like module is likely to be central to their performance, and a similar approach could be applied for controlling its meta-parameters based on numerous modulatory influences. Further studies relating such meta-parameters to other neuromodulatory systems and activation patterns of specific brain areas could provide interesting insights and may prove to be an ultimate test-box for the biological relevance of such an approach.

## Footnotes

[1]TimePreference = (average time between adjacent ITI pokes) / (average response time)

[2]DurationPreference = (average trial response poke duration) / (average ITI poke duration)

## References

[1] J. J. Kim and K. S. Yoon. Stress: metaplastic effects in the hippocampus. *TINS*, 21(12):505–9. 1998.

[2] C. Sandi, M. Loscertales, and C. Guaza. Experience-dependent facilitating effect of corticosterone on spatial memory formation in the water maze. *Eur J Neurosci.*, 9(4):637–42., Apr 1997.

[3] M. Joels, Z. Pu, O. Wiegert, M. S. Oitzl, and H. J. Krugers. Learning under stress: how does it work? *Trends Cogn Sci.*, 10(4):152–8. Apr 2006.

[4] G. Aston-Jones, J. Rajkowski, and J. Cohen. Locus coeruleus and regulation of behavioral flexibility and attention. *Prog Brain Res.*, 126:165–82., 2000.

[5] A. J. Yu and P. Dayan. Uncertainty, Neuromodulation, and Attention. *Neuron*, 46:681–92, May 19 2005.

[6] A. Holmes, C. C. Wrenn, A. P. Harris, K. E. Thayer, and J. N. Crawley. Behavioral profiles of inbred strains on novel olfactory, spatial and emotional tests for reference memory in mice. *Genes Brain Behav.*, 1(1):55–69., Jan 2002.

[7] M. J. Kreek, D. A. Nielsen, E. R. Butelman, and K. S. LaForge. Genetic influences on impulsivity, risk taking, stress responsivity and vulnerability to drug abuse and addiction. *Nat Neurosci.*, 8:1450–7, 2005.

[8] P. Dayan and B. W. Balleine. Reward, Motivation, and Reinforcement Learning. *Neuron*, 36:285–98, 2002.

[9] M. M. Botvinick, T. S. Braver, C. S. Carter, D. M. Barch, and J. D. Cohen. Conflict monitoring and cognitive control. *Psychol Review*,108(3):624–52, Mar 2001.

[10] R. Sutton and A. G. Barto. *Reinforcement Learning - An Introduction*. MIT Press, 1998.

[11] W. Schultz, P. Dayan, and P. R. Montague. A neural substrate of prediction and reward. *Science*, 275(5306):1593–9, Mar 14 1997.

[12] S. C. Tanaka, K. Doya, G. Okada, K. Ueda, Y. Okamoto, and S. Yamawaki. Prediction of immediate and future rewards differentially recruits cortico-basal ganglia loops. *Nat Neurosci.*, 7:887–93, Jul 2004.

[13] A. D. Redish. Addiction as a Computational Process Gone Awry. *Science.*, 306(5703):1944–7, 2004.

[14] K. Doya. Metalearning and neuromodulation. *Neural Netw*, 15(4-6):495–506, Jun-Jul 2002.

[15] K. Doya. Modulators of decision making. *Nat Neurosci.*, 11:410–6, Apr 2008.

[16] Y. Clement, C. Joubert, C. Kopp, E. M. Lepicard, P. Venault, R. Misslin, M. Cadot, and G. Chapouthier Anxiety in Mice: A Principal Component Analysis Study. *Neural Plast.*, 35457, Mar 21 2007.

[17] W. H. Press, B. P. Flannery, S. A. Teukolsky, and W. T. Vetterling. *Numerical Recipes in C : The Art of Scientific Computing*. Cambridge University Press, 1992.

[18] D. Marquardt. An algorithm for least squares estimation of nonlinear parameters. *SIAM J. Appl. Math*, 11:431–441, 1963.

[19] J. Amat, M. V. Baratta, E. Paul, S. T. Bland, L. R. Watkins, and S. F. Maier. Medial prefrontal cortex determines how stressor controllability affects behavior and dorsal raphe nucleus. *Nat Neurosci.*, 8(3):365–71. Mar 2005.

